# ARC-LH: A New Adaptive Resampling Algorithm for Improving ANN Classifiers

**Friedrich Leisch**
Friedrich.Leisch@ci.tuwien.ac.at

**Kurt Hornik**
Kurt.Hornik@ci.tuwien.ac.at

Institut für Statistik und Wahrscheinlichkeitstheorie
Technische Universität Wien
A-1040 Wien, Austria

## Abstract

We introduce arc-lh, a new algorithm for improvement of ANN classifier performance, which measures the importance of patterns by aggregated network output errors. On several artificial benchmark problems, this algorithm compares favorably with other resample and combine techniques.

## 1 Introduction

The training of artificial neural networks (ANNs) is usually a stochastic and unstable process. As the weights of the network are initialized at random and training patterns are presented in random order, ANNs trained on the same data will typically be different in value and performance. In addition, small changes in the training set can lead to two completely different trained networks with different performance even if the nets had the same initial weights.

Roughly speaking, ANNs have a low bias because of their approximation capabilities, but a rather high variance because of the instability. Recently, several resample and combine techniques for improving ANN performance have been proposed. In this paper we introduce an new arcing ("adaptive resample and combine") method called arc-lh. Contrary to the arc-fs method by Freund & Schapire (1995), which uses misclassification rates for adapting the resampling probabilities, arc-lh uses the aggregated network output error. The performance of arc-lh is compared with other techniques on several popular artificial benchmark problems.

## 2   Bias-Variance Decomposition of 0-1 Loss

Consider the task of classifying a random vector $\xi$ taking values in $\mathcal{X}$ into one of $c$ classes $C_1, \ldots, C_c$, and let $g(\cdot)$ be a classification function mapping the input space on the finite set $\{1, \ldots, c\}$.

The classification task is to find an optimal function $g$ minimizing the risk

$$Rg = \mathbb{E}Lg(\xi) = \int_{\mathcal{X}} Lg(x)\,dF(x) \tag{1}$$

where $F$ denotes the (typically unknown) distribution function of $\xi$, and $L$ is a loss function. In this paper, we consider 0-1 loss only, i.e., the loss is 1 for all misclassified patterns and zero otherwise.

It is well known that the optimal classifier, i.e., the classifier with minimum risk, is the Bayes classifier $g^\star$ assigning to each input $x$ the class with maximum posterior probability $\mathbb{P}(C_n|x)$. These posterior probabilities are typically unknown, hence the Bayes classifier cannot be used directly. Note that $Rg^\star = 0$ for disjoint classes and $Rg^\star > 0$ otherwise.

Let $X_N = \{x_1, \ldots, x_N\}$ be a set of independent input vectors for which the true class is known, available for training the classifier. Further, let $g_{X_N}(\cdot)$ denote a classifier trained using set $X_N$. The risk $Rg_{X_N} \geq Rg^\star$ of classifier $g_{X_N}$ is a random variable depending on the training sample $X_N$. In the case of ANN classifiers it also depends on the network training, i.e., even for fixed $X_N$ the performance of a trained ANN is a random variable depending on the initialization of weights and the (often random) presentation of the patterns $[x_n]$ during training.

Following Breiman (1996a) we decompose the risk of a classifier into the (minimum possible) Bayes error, a systematic bias term of the model class and the variance of the classifier within its model class. We call a classifier model *unbiased* for input $x$ if, over replications of all possible training sets $X_N$ of size $N$, network initializations and pattern presentations, $g$ picks the correct class more often than any other class. Let $\mathcal{U} = \mathcal{U}(g)$ denote the set of all $x \in \mathcal{X}$ where $g$ is unbiased; and $\mathcal{B} = \mathcal{B}(g) = \mathcal{X} \backslash \mathcal{U}$ the set of all points where $g$ is biased. The risk of classifier $g$ can be decomposed as

$$Rg = Rg^\star + \text{Bias}(g) + \text{Var}(g) \tag{2}$$

where $Rg^\star$ is the risk of the Bayes classifier,

$$\begin{aligned} \text{Bias}(g) &= R_{\mathcal{B}}g - R_{\mathcal{B}}g^\star \\ \text{Var}(g) &= R_{\mathcal{U}}g - R_{\mathcal{U}}g^\star \end{aligned}$$

and $R_{\mathcal{B}}$ and $R_{\mathcal{U}}$ denote the risk on set $\mathcal{B}$ and $\mathcal{U}$, respectively, i.e., the integration in Equation 1 is over $\mathcal{B}$ or $\mathcal{U}$ instead of $\mathcal{X}$, repectively.

A simpler bias-variance decomposition has been proposed by Kong & Dietterich (1995):

$$\begin{aligned} \text{Bias}(g) &= \mathbb{P}\{\mathcal{B}\} \\ \text{Var}(g) &= Rg - \text{Bias}(g) \end{aligned}$$

The size of the bias set is seen as the bias of the model (i.e., the error the model class "typically" makes). The variance is simply the difference between the actual risk and this bias term. This decompostion yields negative variance if the current classifier performs better than the average classifier.

In both decompositions, the bias gives the systematic risk of the model, whereas the variance measures how good the current realization is compared to the best possible realization of the model. Neural networks are very powerful but rather unstable approximators, hence their bias should be low, but the variance may be high.

# 3    Resample and Combine

Suppose we had $k$ independent training sets $X_{N_1}, \ldots, X_{N_k}$ and corresponding classifiers $g_1, \ldots, g_k$ trained using these sets, respectively. We can then combine these single classifiers into a joint voting classifier $g_k^v$ by assigning to each input $x$ the class the majority of the $g_i$ votes for. If the $g_i$ have low bias, then $g_k^v$ should have low bias, too. If the model is unbiased for an input $x$, then the variance of $g_k^v$ vanishes as $k \to \infty$, and $g^v = \lim_{k \to \infty} g_k^v$ is optimal for $x$. Hence, by resampling training sets from the original training set and combining the resulting classifiers into a voting classifier it might be possible to reduce the high variance of unstable classification algorithms.

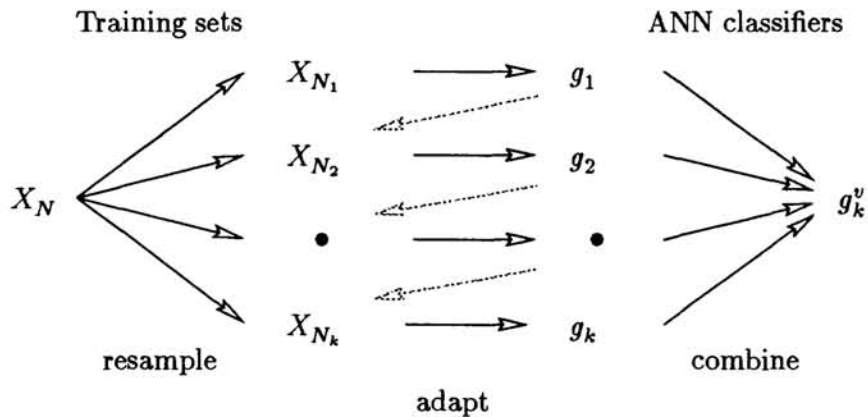

## 3.1    Bagging

Breiman (1994, 1996a) introduced a procedure called bagging ("bootstrap aggregating") for tree classifiers that may also be used for ANNs. The bagging algorithm starts with a training set $X_N$ of size $N$. Several bootstrap replica $X_N^1, \ldots, X_N^k$ are constructed and a neural network is trained on each. These networks are finally combined by majority voting. The bootstrap sets $X_N^i$ consist of $N$ patterns drawn with replacement from the original training set (see Efron & Tibshirani (1993) for more information on the bootstrap).

## 3.2 Arcing

### 3.2.1 Arcing Based on Misclassification Rates

Arcing, which is a more sophisticated version of bagging, was first introduced by Freund & Schapire (1995) and called *boosting*. The new training sets are not constructed by uniformly sampling from the empirical distribution of the training set $X_N$, but from a distribution over $X_N$ that includes information about previous misclassifications.

Let $p_n^i$ denote the probability that pattern $x_n$ is included into the $i$-th training set $X_N^i$ and initialize with $p_n^1 = 1/N$. Freund and Schapire's arcing algorithm, called arc-fs as in Breiman (1996a), works as follows:

1. Construct a pattern set $X_N^i$ by sampling with replacement with probabilities $p_n^i$ from $X_N$ and train a classifier $g_i$ using set $X_N^i$.

2. Set $d_n = 1$ for all patterns that are misclassified by $g_i$ and zero otherwise. With $\epsilon_i = \sum_{n=1}^{N} p_n^i d_n$ and $\beta_i = (1 - \epsilon_i)/\epsilon_i$ update the probabilities by

$$p_n^{i+1} = \frac{p_n^i \beta_i^{d_n}}{\sum_{n=1}^{N} p_n^i \beta_i^{d_n}}$$

3. Set $i := i + 1$ and repeat.

After $k$ steps, $g_1, \ldots, g_k$ are combined with weighted voting were each $g_i$'s vote has weight $\log \beta_i$. Breiman (1996a) and Quinlan (1996) compare bagging and arcing for CART and C4.5 classifiers, respectively. Both bagging and arc-fs are very effective in reducing the high variance component of tree classifiers, with adaptive resampling being a bit better than simple bagging.

### 3.2.2 Arcing Based on Network Error

Independently from the arcing and bagging procedures described above, adaptive resampling has been introduced for active pattern selection in leave-$k$-out cross-validation CV/APS (Leisch & Jain, 1996; Leisch et al., 1995). Whereas arc-fs (or Breiman's arc-x4) uses only the information whether a pattern is misclassified or not, in CV/APS the fact that MLPs approximate the posterior probabilities of the classes (Kanaya & Miyake, 1991) is utilized, too. We introduce a simple new arcing method based on the main idea of CV/APS that the "importance" of a pattern for the learning process can be measured by the aggregated output error of an MLP for the pattern over several training runs.

Let the classifier $g$ be an ANN using 1-of-$c$ coding, i.e., one output node per class, the target $t(x)$ for each input $x$ is one at the node corresponding to the class of $x$ and zero at the remaining output nodes. Let $e(x) = |t(x) - g(x)|^2$ be the squared error of the network for input $x$. Patterns that repeatedly have high output errors are somewhat harder to learn for the network and therefore their resampling probabilities are increased proportionally to the error. Error-dependent resampling

introduces a "grey-scale" of pattern-importance as opposed to the "black and white" paradigm of misclassification dependent resampling.

Again let $p_n^i$ denote the probability that pattern $x_n$ is included into the $i$-th training set $X_N^i$ and initialize with $p_n^1 = 1/N$. Our new arcing algorithm, called arc-lh, works as follows:

1. Construct a pattern set $X_N^i$ by sampling with replacement with probabilities $p_n^i$ from $X_N$ and train a classifier $g_i$ using set $X_N^i$.

2. Add the network output error of each pattern to the resampling probabilities:

$$ p_n^{i+1} = \frac{p_n^i + e_i(x_n)}{\sum_{n=1}^N p_n^i + e_i(x_n)}, \qquad e_i(x_n) = |t(x_n) - g_i(x_n))|^2 $$

3. Set $i := i + 1$ and repeat.

After $k$ steps, $g_1, \ldots, g_k$ are combined by majority voting.

## 3.3   Jittering

In our experiments, we also compare the above resample and combine methods with jittering, which resamples the training set by contaminating the inputs by artificial noise. No voting is done, but the size of the training set is increased by creation of artificial inputs "around" the original inputs, see Koistinen & Holmström (1992).

## 4   Experiments

We demonstrate the effects of bagging and arcing on several well known artificial benchmark problems. For all problems, $i - h - c$ single hidden layer perceptrons (SHLPs) with $i$ input, $h$ hidden and $c$ output nodes were used. The number of hidden nodes $h$ was chosen in a way that the corresponding networks have reasonably low bias.

**2 Spirals with noise:** 2-dimensional input, 2 classes. Inputs with uniform noise around two spirals. $N = 300$. $Rg^\star = 0\%$. 2-14-2 SHLP.

**Continuous XOR:** 2-dimensional input, 2 classes. Uniform inputs on the 2-dimensional square $-1 \leq x, y \leq 1$ classified in the two classes $x * y \geq 0$ and $x * y < 0$. $N = 300$. $Rg^\star = 0\%$. 2-4-2 SHLP.

**Ringnorm:** 20-dimensional input, 2 classes. Class 1 is normal wit mean zero and covariance 4 times the identity matrix. Class 2 is a unit normal with mean $(a, a, \ldots, a)$. $a = 2/\sqrt{20}$. $N = 300$. $Rg^\star = 1.2\%$. 20-4-2 SHLP.

The first two problems are standard benchmark problems (note however that we use a noisy variant of the standard spirals problem); the last one is, e.g., used in Breiman (1994, 1996a).

All experiments were replicated 50 times, in each bagging and arcing replication
10 classifiers were combined to build a voting classifier. Generalization errors were
computed using Monte Carlo techniques on test sets of size 10000.

Table 1 gives the average risk over the 50 replications for a standard single SHLP,
an SHLP trained on a jittered training set and for voting classifiers using ten votes
constructed with bagging, arc-lh and arc-fs, respectively. The Bayes risk of the spiral
and xor example is zero, hence the risk of a network equals the sum of its bias and
variance. The Bayes risk of the ringnorm example is 1.2%.

| | $Rg$ | Breiman Bias($g$) | Var($g$) | Kong & Dietterich Bias($g$) | Var($g$) |
|---|---|---|---|---|---|
| | | 2 Spirals | | | |
| standard | 7.75 | 0.32 | 7.43 | 0.82 | 6.93 |
| jitter | 6.53 | 0.26 | 6.27 | 0.52 | 6.02 |
| bagging | 4.39 | 0.35 | 4.04 | 0.68 | 3.71 |
| arc-fs | 4.31 | 0.35 | 3.96 | 0.60 | 3.71 |
| arc-lh | 4.32 | 0.31 | 4.01 | 0.72 | 3.60 |
| | | XOR | | | |
| standard | 6.54 | 0.53 | 6.01 | 1.32 | 5.22 |
| jitter | 6.29 | 0.37 | 5.92 | 1.08 | 5.21 |
| bagging | 3.69 | 0.59 | 3.09 | 1.22 | 2.47 |
| arc-fs | 3.73 | 0.58 | 3.15 | 1.12 | 2.61 |
| arc-lh | 3.58 | 0.50 | 3.08 | 1.20 | 2.38 |
| | | Ringnorm | | | |
| standard | 18.64 | 9.19 | 8.26 | 13.84 | 4.80 |
| jitter | 18.56 | 9.03 | 8.34 | 13.72 | 4.84 |
| bagging | 15.72 | 9.61 | 4.91 | 13.54 | 2.18 |
| arc-fs | 15.71 | 9.70 | 4.81 | 13.58 | 2.13 |
| arc-lh | 15.63 | 9.30 | 5.13 | 13.20 | 2.43 |

Table 1: Bias-variance decompositions.

The variance part was drastically reduced by the resample & combine methods, with
only a negligible change in bias. Note the low bias in the spiral and xor problems.
ANNs obviously can solve these classification tasks (one could create appropriate
nets by hand), but of course training cannot find the exact boundaries between the
classes. Averaging over several nets helps to overcome this problem. The bias in
the ringnorm example is rather high, indicating that a change of network topology
(bigger net, etc.) or training algorithm (learning rate, etc.) may lower the overall
risk.

## 5  Summary

Comparison of of the resample and combine algorithms shows slight advantages
for adaptive resampling, but no algorithm dominates the other two. Further im-

provements should be possible based on a better understanding of the theoretical properties of resample and combine techniques. These issues are currently being investigated.

# References

Breiman, L. (1994). *Bagging predictors.* Tech. Rep. 421, Department of Statistics, University of California, Berkeley, California, USA.

Breiman, L. (1996a). *Bias, variance, and arcing classifiers.* Tech. Rep. 460, Statistics Department, University of California, Berkeley, CA, USA.

Breiman, L. (1996b). Stacked regressions. *Machine Learning*, **24**, 49.

Drucker, H. & Cortes, C. (1996). Boosting decision trees. In Touretzky, S., Mozer, M. C., & Hasselmo, M. E. (eds.), *Advances in Neural Information Processing Systems*, vol. 8. MIT Press.

Efron, B. & Tibshirani, R. J. (1993). *An introduction to the bootstrap.* Monographs on Statistics and Applied Probability. New York: Chapman & Hall.

Freund, Y. & Schapire, R. E. (1995). *A decision-theoretic generalization of on-line learning and an application to boosting.* Tech. rep., AT&T Bell Laboratories, 600 Mountain Ave, Murray Hill, NJ, USA.

Kanaya, F. & Miyake, S. (1991). Bayes statistical behavior and valid generalization of pattern classifying neural networks. *IEEE Transactions on Neural Networks*, **2**(4), 471–475.

Kohavi, R. & Wolpert, D. H. (1996). Bias plus variance decomposition for zero-one loss. In *Machine Learning: Proceedings of the 13th International Conference*.

Koistinen, P. & Holmström, L. (1992). Kernel regression and backpropagation training with noise. In Moody, J. E., Hanson, S. J., & Lippmann, R. P. (eds.), *Advances in Neural Information Processing Systems*, vol. 4, pp. 1033–1039. Morgan Kaufmann Publishers, Inc.

Kong, E. B. & Dietterich, T. G. (1995). Error-correcting output coding corrects bias and variance. In *Machine Learning: Proceedings of the 12th International Conference*, pp. 313–321. Morgan-Kaufmann.

Leisch, F. & Jain, L. C. (1996). Cross-validation with active pattern selection for neural network classifiers. Submitted to IEEE Transactions on Neural Networks, in Review.

Leisch, F., Jain, L. C., & Hornik, K. (1995). NN classifiers: Reducing the computational cost of cross-validation by active pattern selection. In *Artificial Neural Networks and Expert Systems*, vol. 2. Los Alamitos, CA, USA: IEEE Computer Society Press.

Quinlan, J. R. (1996). Bagging, boosting and C4.5. University of Sydney, Australia.

Ripley, B. D. (1996). *Pattern recognition and neural networks.* Cambridge, UK: Cambridge University Press.

Tibshirani, R. (1996a). Bias, variance and prediction error for classification rules. University of Toronto, Canada.

Tibshirani, R. (1996b). A comparison of some error estimates for neural network models. *Neural Computation*, **8**(1), 152–163.